# Exponential Concentration for Mutual Information Estimation with Application to Forests

**Han Liu**
Department of Operations Research
and Financial Engineering
Princeton University, NJ 08544
hanliu@princeton.edu

**John Lafferty**
Department of Computer Science
Department of Statistics
University of Chicago, IL 60637
lafferty@galton.uchicago.edu

**Larry Wasserman**
Department of Statistics
Machine Learning Department
Carnegie Mellon University, PA 15231
larry@stat.cmu.edu

## Abstract

We prove a new exponential concentration inequality for a plug-in estimator of the Shannon mutual information. Previous results on mutual information estimation only bounded expected error. The advantage of having the exponential inequality is that, combined with the union bound, we can guarantee accurate estimators of the mutual information for many pairs of random variables simultaneously. As an application, we show how to use such a result to optimally estimate the density function and graph of a distribution which is Markov to a forest graph.

## 1  Introduction

We consider the problem of nonparametrically estimating the Shannon mutual information between two random variables. Let $X_1 \in \mathcal{X}_1$ and $X_2 \in \mathcal{X}_2$ be two random variables with domains $\mathcal{X}_1$ and $\mathcal{X}_2$ and joint density $p(x_1, x_2)$. The mutual information between $X_1$ and $X_2$ is

$$I(X_1; X_2) := \int_{\mathcal{X}_1} \int_{\mathcal{X}_2} p(x_1, x_2) \log \left( \frac{p(x_1, x_2)}{p(x_1)p(x_2)} \right) dx_1 \, dx_2 = H(X_1) + H(X_2) - H(X_1, X_2),$$

where $H(X_1, X_2) = -\int \int p(x_1, x_2) \log p(x_1, x_2) dx_1 \, dx_2$ (and similarly for $H(X_1)$ and $H(X_2)$) are the corresponding Shannon entropies [4]. The mutual information is a measure of dependence between $X_1$ and $X_2$. To estimate $I(X_1; X_2)$ well, it suffices to estimate $H(X_1, X_2) := H(p)$.

A simple way to estimate the Shannon entropy is to use a kernel density estimator (KDE) [22, 1, 9, 5, 20, 7], i.e., the densities $p(x, y)$, $p(x)$, and $p(y)$ are separately estimated from samples and the estimated densities are used to calculate the entropy. Alternative methods involve estimation of the entropies using spacings [25, 26, 23], k-nearest neighbors [11, 12], the Edgeworth expansion [24], and convex optimization [17]. More discussions can be found in the survey articles [2, 19]. There have been many recent developments in the problem of estimating Shannon entropy and related quantities as well as application of these results to machine learning problems [18, 21, 8, 6]. Under weak conditions, it has been shown that there are estimators that achieve the parametric $\sqrt{n}$-rate of convergence in mean squared error (MSE), where $n$ is the sample size.

In this paper, we construct an estimator with this rate, but we also prove an exponential concentration inequality for the estimator. More specifically, we show that our estimator $\widehat{H}$ of $H(p)$ satisfies

$$\sup_{p \in \Sigma} \mathbb{P} \left( |\widehat{H} - H(p)| > \epsilon \right) \leq 2 \exp \left( -\frac{n\epsilon^2}{36\kappa^2} \right) \tag{1.1}$$

where $\Sigma$ is a nonparametric class of distributions defined in Section 2 and $\kappa$ is a constant. To the best of our knowledge, this is the first such exponential inequality for nonparametric Shannon entropy and mutual information estimation. The advantage of this result, over the usual results which state that $\mathbb{E}\big(|\widehat{H} - H(p)|^2\big) = O(n^{-1})$, is that we can apply the union bound and thus guarantee accurate mutual information estimation for many pairs of random variables simultaneously. As an application, we consider forest density estimation [15], which, in a $d$-dimenionsal problem, requires estimating $\frac{d(d+1)}{2}$ mutual informations in order to apply the Chow-Liu algorithm. As long as $\frac{\log d}{n} \to 0$ as $n \to \infty$, we can estimate the forest graph well, even if $d = d(n)$ increases with $n$ exponentially fast.

The rest of this paper is organized as follows. The assumptions and estimator are given in Section 2. The main theoretical analysis is in Section 3. In Section 4 we show how to apply the result to forest density estimation. Some discussion and possible extensions are provided in the last section.

## 2  Estimator and Main Result

Let $X = (X_1, X_2) \in \mathbb{R}^2$ be a random vector with density $p(x) := p(x_1, x_2)$ and let $x^1, \ldots, x^n \in \mathcal{X} \subset \mathbb{R}^2$ be a random sample from $p$. In this paper, we only consider the case of bounded domain $\mathcal{X} = [0, 1]^2$. We want to estimate the Shannon entropy

$$H(p) = -\int_{\mathcal{X}} p(x) \log p(x) dx. \tag{2.1}$$

We start with some assumptions on the density function $p(x_1, x_2)$.

**Assumption 2.1 (Density assumption).** We assume the density $p(x_1, x_2)$ belongs to a 2nd-order Hölder class $\Sigma_\kappa(2, L)$ and is bounded away from zero and infinity. In particular, there exist constants $\kappa_1, \kappa_2$

$$0 < \kappa_1 \leq \min_{x \in \mathcal{X}} p(x) \leq \max_{x \in \mathcal{X}} p(x) \leq \kappa_2 < \infty, \tag{2.2}$$

and for any $(x_1, x_2)^T \in \mathcal{X}$, there exists a constant $L$ such that, for any $(u, v)^T \in \mathcal{X}$

$$\left| p(x_1 + u, x_2 + v) - p(x_1, x_2) - \frac{\partial p(x_1, x_2)}{\partial x_1} u - \frac{\partial p(x_1, x_2)}{\partial x_2} v \right| \leq L(u^2 + v^2). \tag{2.3}$$

**Assumption 2.2 (Boundary assumption).** If $\{x_n\} \in \mathcal{X}$ is any sequence converging to a boundary point $x^*$, we require the density $p(x)$ has vanishing first order partial derivatives:

$$\lim_{n \to \infty} \frac{\partial p(x_n)}{\partial x_1} = \lim_{n \to \infty} \frac{\partial p(x_n)}{\partial x_2} = 0. \tag{2.4}$$

To efficiently estimate the entropy in (2.1), we use a KDE based "plug-in" estimator. Bias at the boundaries turns out to be very important in this problem; see [10] for a discussion of boundary bias. To correct the boundary effects, we use the following "mirror image" kernel density estimator:

$$
\begin{aligned}
\widetilde{p}_h(x_1, x_2) := \frac{1}{nh^2} \sum_{i=1}^{n} \Bigg\{ &K\left(\frac{x_1 - x_1^i}{h}\right) K\left(\frac{x_2 - x_2^i}{h}\right) + K\left(\frac{x_1 + x_1^i}{h}\right) K\left(\frac{x_2 - x_2^i}{h}\right) \\
&+ K\left(\frac{x_1 - x_1^i}{h}\right) K\left(\frac{x_2 + x_2^i}{h}\right) + K\left(\frac{x_1 + x_1^i}{h}\right) K\left(\frac{x_2 + x_2^i}{h}\right) \\
+ K\left(\frac{x_1 - x_1^i}{h}\right) K\left(\frac{x_2 - 2 + x_2^i}{h}\right) &+ K\left(\frac{x_1 + x_1^i}{h}\right) K\left(\frac{x_2 - 2 + x_2^i}{h}\right) \\
+ K\left(\frac{x_1 - 2 + x_1^i}{h}\right) K\left(\frac{x_2 - x_2^i}{h}\right) &+ K\left(\frac{x_1 - 2 + x_1^i}{h}\right) K\left(\frac{x_2 + x_2^i}{h}\right) \\
&+ K\left(\frac{x_1 - 2 + x_1^i}{h}\right) K\left(\frac{x_2 - 2 + x_2^i}{h}\right) \Bigg\}. \tag{2.5}
\end{aligned}
$$

Here $h$ is the bandwidth and $K(\cdot)$ is a univariate kernel function. We denote by $K_2(u, v) := K(u)K(v)$ the bivariate product kernel. This estimator has nine terms; one corresponds to the original data in the unit square $[0, 1]^2$, and each of the remaining terms corresponds to reflecting the data across one of the four sides or four corners of the square.

**Assumption 2.3 (Kernel assumption).** The kernel $K(\cdot)$ is nonnegative and has a bounded support $[-1, 1]$ with $\int_{-1}^{1} K(u)du = 1$ and $\int_{-1}^{1} uK(u)du = 0$.

By Assumption 2.1, the values of the true density lie in the interval $[\kappa_1, \kappa_2]$. We propose a clipped KDE estimator

$$\widehat{p}_h(x) = T_{\kappa_1, \kappa_2}\left(\widetilde{p}_h(x)\right), \tag{2.6}$$

where $T_{\kappa_1, \kappa_2}(a) = \kappa_1 \cdot I(a < \kappa_1) + a \cdot I(\kappa_1 \le a \le \kappa_2) + \kappa_2 \cdot I(a > \kappa_2)$, so that the estimated density also has this property. Letting $g(u) = u \log u$, we propose the following plug-in entropy estimator:

$$H\left(\widehat{p}_h\right) := -\int_{\mathcal{X}} g\left(\widehat{p}_h(x)\right) dx = -\int_{\mathcal{X}} \widehat{p}_h(x) \log \widehat{p}_h(x) dx. \tag{2.7}$$

**Remark 2.1.** The clipped estimator $\widehat{p}_h$ requires the knowledge of $\kappa_1$ and $\kappa_2$. In applications, we do not need to know the exact values of $\kappa_1$ and $\kappa_2$; lower and upper bounds are sufficient.

Our main technical result is the following exponential concentration inequality on $H(\widehat{p}_h)$ around the population quantity $H(p)$. Our proof is given in Section 3.

**Theorem 2.1.** *Under Assumptions 2.1, 2.2, and 2.3, if we choose the bandwidth according to $h \asymp n^{-1/4}$, then there exists a constant $N_0$ such that for all $n > N_0$,*

$$\sup_{p \in \Sigma_\kappa(2, L)} \mathbb{P}\left(|H\left(\widehat{p}_h\right) - H\left(p\right)| > \epsilon\right) \le 2 \exp\left(-\frac{n\epsilon^2}{36\kappa^2}\right), \tag{2.8}$$

*where $\kappa = \max\{|\log \kappa_1|, |\log \kappa_2|\} + 1$.*

To the best of our knowledge, this is the first time an exponential inequality like (2.8) has been established for Shannon entropy estimation over the Hölder class. It is easy to see that (2.8) implies the parametric $\sqrt{n}$-rate of convergence in mean squared error, $\mathbb{E}(|\widehat{H} - H(p)|) = O(n^{-1/2})$. The bandwidth $h \asymp n^{-1/4}$ in the above theorem is different from the usual choice for optimal bivariate density estimation, which is $h_P \asymp n^{-1/6}$ for the 2nd-order Hölder class. By using $h \asymp n^{-1/4}$, we undersmooth the density estimate. As we show in the next section, such a bandwidth choice is important for achieving the optimal rate for entropy estimation.

Let $I(p) := I(X_1; X_2)$ be the Shannon mutual information, and define

$$I(\widehat{p}_h) := \int_{\mathcal{X}_1} \int_{\mathcal{X}_2} \widehat{p}_h(x_1, x_2) \log\left(\frac{\widehat{p}_h(x_1, x_2)}{\widehat{p}_h(x_1)\widehat{p}_h(x_2)}\right) dx_1 \, dx_2. \tag{2.9}$$

The next corollary provides an exponential inequality for Shannon mutual information estimation.

**Corollary 2.1.** *Under the same conditions as in Theorem 2.1, if we choose $h \asymp n^{-1/4}$, then there exists a constant $N_1$, such that for all $n > N_1$,*

$$\sup_{p \in \Sigma_\kappa(2, L)} \mathbb{P}\left(|I\left(\widehat{p}_h\right) - I\left(p\right)| > \epsilon\right) \le 6 \exp\left(-\frac{n\epsilon^2}{324\kappa^2}\right), \tag{2.10}$$

*where $\kappa = \max\{|\log \kappa_1|, |\log \kappa_2|\} + 1$.*

*Proof.* Using the same proof for Theorem 2.1, we can show that (2.8) also holds for estimating univariate entropies $H(X_1)$ and $H(X_2)$. The desired result then follows from the union bound since $I(p) := I(X_1; X_2) = H(X_1) + H(X_2) - H(X_1, X_2)$. □

**Remark 2.2.** We use the same bandwidth $h \asymp n^{-1/4}$ to estimate the bivariate density $p(x_1, x_2)$ and univariate densities $p(x_1), p(x_2)$. A related result is presented in [15]. They consider the same problem setting as ours and also use a KDE based plug-in estimator to estimate the mutual information. However, unlike our proposal, they advocate the use of different bandwidths for bivariate and univariate entropy estimations. For bivariate case they use $h_2 \asymp n^{-1/6}$; for univariate case they use $h_1 \asymp n^{-1/5}$. Such bandwidths $h_1$ and $h_2$ are useful for optimally estimating the density functions. However, such a choice achieves a suboptimal rate in terms of mutual information estimation: $\sup_{p \in \Sigma_\kappa(2, L)} \mathbb{P}\left(|I\left(\widehat{p}_h\right) - I\left(p\right)| > \epsilon\right) \le c_1 \exp\left(-c_2 n^{2/3} \epsilon^2\right)$, where $c_1$ and $c_2$ are two constants. Our method achieves the faster parametric rate.

# 3 Theoretical Analysis

Here we present the detailed proof of Theorem 2.1. To analyze the error $|H(\widehat{p}_h) - H(p)|$, we first decompose it into a bias or approximation error term, and a "variance" or estimation error term:

$$|H(\widehat{p}_h) - H(p)| \le \underbrace{|H(\widehat{p}_h) - \mathbb{E}H(\widehat{p}_h)|}_{\text{Variance}} + \underbrace{|\mathbb{E}H(\widehat{p}_h) - H(p)|}_{\text{Bias}}. \tag{3.1}$$

We are going to show that

$$\sup_{p \in \Sigma_\kappa(2,L)} \mathbb{P}\Big(\underbrace{|H(\widehat{p}_h) - \mathbb{E}H(\widehat{p}_h)|}_{\text{Variance}} > \epsilon\Big) \quad \le \quad 2\exp\left(-\frac{n\epsilon^2}{32\kappa^2}\right), \tag{3.2}$$

$$\sup_{p \in \Sigma_\kappa(2,L)} \underbrace{|\mathbb{E}H(\widehat{p}_h) - H(p)|}_{\text{Bias}} \quad \le \quad c_1 h^2 + \frac{c_3}{nh^2}, \tag{3.3}$$

where $c_1$ and $c_3$ are two constants. Since the bound on the variance in (3.2) does not depend on $h$, to optimize the rate, we only need to choose $h$ to minimize the righthand side of (3.3). Therefore $h \asymp n^{-1/4}$ achieves the optimal rate. In the rest of this section, we bound the bias and variance terms separately.

## 3.1 Analyzing the Bias Term

Here we prove (3.3). Let $u$ be a vector. We denote the sup norm by $\|u\|_\infty$. The next lemma bounds the integrated squared bias of the kernel density estimator over the support $\mathcal{X} := [0,1]^2$.

**Lemma 3.1.** *Under Assumptions 2.1, 2.2, and 2.3, there exists a constant $c > 0$ such that*

$$\sup_{p \in \Sigma_\kappa(2,L)} \int_\mathcal{X} (\mathbb{E}\widetilde{p}_h(x) - p(x))^2 \, dx \le ch^4. \tag{3.4}$$

*Proof.* We partition the support $\mathcal{X} := [0,1]^2$ into three regions $\mathcal{X} = \mathcal{B} \cup \mathcal{C} \cup \mathcal{I}$, the boundary area $\mathcal{B}$, the corner area $\mathcal{C}$, and the interior area $\mathcal{I}$:

$$\mathcal{C} \;=\; \{x : \|x - u\|_\infty \le h \text{ for } u = (0,0)^T,\ \text{or } (0,1)^T,\ \text{or } (1,0)^T,\ \text{or } (1,1)^T\}, \tag{3.5}$$
$$\mathcal{B} \;=\; \{x : x \text{ is within distance } h \text{ to an edge of } \mathcal{X},\ \text{but does not belong to } \mathcal{C}\}, \tag{3.6}$$
$$\mathcal{I} \;=\; \mathcal{X} \setminus (\mathcal{C} \cup \mathcal{B}). \tag{3.7}$$

We have the following decomposition:

$$\int_\mathcal{X} (\mathbb{E}\widetilde{p}_h(x) - p(x))^2 \, dx = \int_\mathcal{I} + \int_\mathcal{C} + \int_\mathcal{B} (\mathbb{E}\widetilde{p}_h(x) - p(x))^2 \, dx = T_\mathcal{I} + T_\mathcal{C} + T_\mathcal{B}.$$

From standard results on kernel density estimation, we know that $\sup_{p \in \Sigma(2,L)} T_\mathcal{I} \le ch^4$. In the next two subsections, we bound $T_\mathcal{B} := \int_\mathcal{B} (\mathbb{E}\widetilde{p}_h(x) - p(x))^2 \, dx$ and $T_\mathcal{C} := \int_\mathcal{C} (\mathbb{E}\widetilde{p}_h(x) - p(x))^2 \, dx$.

### 3.1.1 Analyzing $T_\mathcal{B}$

Let $\mathcal{A} := \{x : 0 \le x_1 \le h \text{ and } h \le x_2 \le 1 - h\}$. We have

$$T_\mathcal{B} \;=\; \int_\mathcal{B} (\mathbb{E}\widetilde{p}_h(x) - p(x))^2 \, dx \le c \int_\mathcal{A} (\mathbb{E}\widetilde{p}_h(x) - p(x))^2 \, dx. \tag{3.8}$$

For $x \in \mathcal{A}$, we have

$$\widetilde{p}_h(x) = \frac{1}{nh^2} \sum_{i=1}^n \left[ K\left(\frac{x_1 - x_1^i}{h}\right) K\left(\frac{x_2 - x_2^i}{h}\right) + K\left(\frac{x_1 + x_1^i}{h}\right) K\left(\frac{x_2 - x_2^i}{h}\right) \right]. \tag{3.9}$$

Therefore, for $x \in \mathcal{A}$ we have

$$\mathbb{E}\widetilde{p}_h(x) \;=\; \frac{1}{h^2} \int_0^1 \int_0^1 K\left(\frac{x_1 - t_1}{h}\right) K\left(\frac{x_2 - t_2}{h}\right) p(t_1, t_2) \, dt_1 dt_2$$

$$+\frac{1}{h^2}\int_0^1\int_0^1 K\left(\frac{x_1+t_1}{h}\right)K\left(\frac{x_2-t_2}{h}\right)p(t_1,t_2)dt_1dt_2$$

$$= \int_{-1}^1\int_{-\frac{x_1}{h}}^1 K(u_1)K(u_2)p(x_1+u_1h,x_2+u_2h)du_1du_2$$

$$+\int_{-1}^1\int_{-1}^{-\frac{x_1}{h}} K(u_1)K(u_2)p(x_1-u_1h,x_2-u_2h)du_1du_2. \tag{3.10}$$

Since $p\in\Sigma_\kappa(2,L)$ and $0<x_1\le h$, we have

$$|p(x_1+u_1h,x_2+u_2h)-p(x_1,x_2)-\langle\nabla p(x),u\rangle h|\le L\|u\|_2^2h^2,$$

$$\left|p(x_1-u_1h,x_2-u_2h)-p(x_1,x_2)+\frac{\partial p(x)}{\partial x_1}(2x_1+u_1h)+\frac{\partial p(x)}{\partial x_2}(u_2h)\right|\le L[(2+u_1)^2+u_2^2]h^2.$$

Since $|u_1|,|u_2|\le 1$, we have $|p(x_1+u_1h,x_2+u_2h)-p(x_1,x_2)|\le\left|\frac{\partial p(x)}{\partial x_1}\right|h+\left|\frac{\partial p(x)}{\partial x_2}\right|h+$

$L\|u\|_2^2h^2$. Similarly, $|p(x_1-u_1h,x_2-u_2h)-p(x_1,x_2)|\le 9\left|\frac{\partial p(x)}{\partial x_1}\right|h+\left|\frac{\partial p(x)}{\partial x_2}\right|h+10Lh^2$.

For any $x\in\mathcal{A}$, we can bound the bias term

$$|\mathbb{E}\widetilde{p}_h(x)-p(x)| \tag{3.11}$$

$$= \left|\mathbb{E}\widetilde{p}_h(x)-\int_{-1}^1\int_{-1}^1 K(u_1)K(u_2)p(t_1,t_2)du_1du_2\right| \tag{3.12}$$

$$\le \int_{-1}^1\int_{-\frac{x_1}{h}}^1 K(u_1)K(u_2)\big|p(x_1+u_1h,x_2+u_2h)-p(x_1,x_2)\big|du_1du_2 \tag{3.13}$$

$$+\int_{-1}^1\int_{-1}^{-\frac{x_1}{h}} K(u_1)K(u_2)\big|p(-u_1h-x_1,x_2-u_2h)-p(x_1,x_2)\big|du_1du_2 \tag{3.14}$$

$$\le 10\left|\frac{\partial p(x)}{\partial x_1}\right|h+2\left|\frac{\partial p(x)}{\partial x_2}\right|h+12Lh^2$$

$$\le 12Lh^2+12Lh^2$$

$$= 24Lh^2,$$

where the last inequality follows from the fact that $\left|\frac{\partial p(x)}{\partial x_1}\right|,\left|\frac{\partial p(x)}{\partial x_2}\right|\le Lh$, by the Hölder condition and the assumption that the density $p(x)$ has vanishing partial derivatives on the boundary points. Therefore, we have $T_\mathcal{B}\le ch^5$.

### 3.1.2 Analyzing $T_\mathcal{C}$

Let $\mathcal{A}_1:=\{x:0\le x_1,x_2\le h\}$. We now analyze the term $T_\mathcal{C}$:

$$T_\mathcal{C}=\int_\mathcal{C}(\mathbb{E}\widetilde{p}_h(x)-p(x))^2\,dx\le c\int_{\mathcal{A}_1}(\mathbb{E}\widetilde{p}_h(x)-p(x))^2\,dx. \tag{3.15}$$

For notational simplicity, we write

$$U_{x,h}(a,b)=K\left(\frac{x_1-a}{h}\right)K\left(\frac{x_2-b}{h}\right). \tag{3.16}$$

For $x\in\mathcal{A}_1$, we have

$$\widetilde{p}_h(x)=\frac{1}{nh^2}\sum_{i=1}^n\left[U_{x,h}(x_1^i,x_2^i)+U_{x,h}(-x_1^i,x_2^i)+U_{x,h}(x_1^i,-x_2^i)+U_{x,h}(-x_1^i,-x_2^i)\right]. \tag{3.17}$$

Therefore, for $x\in\mathcal{A}_1$ we have

$$\mathbb{E}\widetilde{p}_h(x)$$

$$= \frac{1}{h^2}\int_0^1\int_0^1\left[U_{x,h}(t_1,t_2)+U_{x,h}(-t_1,t_2)+U_{x,h}(t_1,-t_2)+U_{x,h}(-t_1,-t_2)\right]p(t_1,t_2)dt_1dt_2$$

$$= \int_{-\frac{x_2}{h}}^{1} \int_{-\frac{x_1}{h}}^{1} K(u_1)K(u_2)p(x_1 + u_1 h, x_2 + u_2 h)du_1 du_2 \tag{3.18}$$

$$+ \int_{-\frac{x_2}{h}}^{1} \int_{\frac{x_1}{h}}^{1} K(u_1)K(u_2)p(u_1 h - x_1, x_2 + u_2 h)du_1 du_2 \tag{3.19}$$

$$+ \int_{\frac{x_2}{h}}^{1} \int_{-\frac{x_1}{h}}^{1} K(u_1)K(u_2)p(u_1 h + x_1, -x_2 + u_2 h)du_1 du_2 \tag{3.20}$$

$$+ \int_{\frac{x_2}{h}}^{1} \int_{\frac{x_1}{h}}^{1} K(u_1)K(u_2)p(-x_1 + u_1 h, -x_2 + u_2 h)du_1 du_2. \tag{3.21}$$

Since $K(\cdot)$ is a symmetric kernel on $[-1, 1]$, we have

$$\int_{-\frac{x_2}{h}}^{1} \int_{\frac{x_1}{h}}^{1} K(u_1)K(u_2)du_1 du_2 = \int_{-\frac{x_2}{h}}^{1} \int_{-1}^{-\frac{x_1}{h}} K(u_1)K(u_2)du_1 du_2, \tag{3.22}$$

$$\int_{\frac{x_2}{h}}^{1} \int_{-\frac{x_1}{h}}^{1} K(u_1)K(u_2)du_1 du_2 = \int_{-1}^{-\frac{x_2}{h}} \int_{-\frac{x_1}{h}}^{1} K(u_1)K(u_2)du_1 du_2. \tag{3.23}$$

Therefore, for $x = (x_1, x_2)^T \in \mathcal{A}_1$,

$$p(x_1, x_2) = \int_{-\frac{x_2}{h}}^{1} \int_{-\frac{x_1}{h}}^{1} + \int_{-\frac{x_2}{h}}^{1} \int_{\frac{x_1}{h}}^{1} + \int_{\frac{x_2}{h}}^{1} \int_{-\frac{x_1}{h}}^{1} + \int_{\frac{x_2}{h}}^{1} \int_{\frac{x_1}{h}}^{1} p(x_1, x_2)K(u_1)K(u_2)du_1 du_2.$$

Using the fact that $p \in \Sigma_\kappa(2, L)$, $0 \le x_1, x_2 \le h$, and $-1 \le u_1, u_2 \le 1$, we have

$$|p(x_1 + u_1 h, x_2 + u_2 h) - p(x_1, x_2)| \le 4Lh^2, \tag{3.24}$$

$$|p(u_1 h - x_1, x_2 + u_2 h) - p(x_1, x_2)| \le 20Lh^2, \tag{3.25}$$

$$|p(u_1 h + x_1, u_2 h - x_2) - p(x_1, x_2)| \le 20Lh^2, \tag{3.26}$$

$$|p(u_1 h - x_1, u_2 h - x_2) - p(x_1, x_2)| \le 36Lh^2. \tag{3.27}$$

For $x \in \mathcal{A}_1$, we can then bound the bias term as

$$|\mathbb{E}\widetilde{p}_h(x) - p(x)| \tag{3.28}$$

$$= \left| \mathbb{E}\widetilde{p}_h(x) - \int_{-1}^{1} \int_{-1}^{1} K(u_1)K(u_2)p(t_1, t_2)du_1 du_2 \right| \tag{3.29}$$

$$\le \int_{-\frac{x_2}{h}}^{1} \int_{-\frac{x_1}{h}}^{1} K(u_1)K(u_2)|p(x_1 + u_1 h, x_2 + u_2 h) - p(x_1, x_2)|du_1 du_2 \tag{3.30}$$

$$+ \int_{-\frac{x_2}{h}}^{1} \int_{\frac{x_1}{h}}^{1} K(u_1)K(u_2)|p(u_1 h - x_1, x_2 + u_2 h) - p(x_1, x_2)|du_1 du_2 \tag{3.31}$$

$$+ \int_{\frac{x_2}{h}}^{1} \int_{-\frac{x_1}{h}}^{1} K(u_1)K(u_2)|p(u_1 h + x_1, u_2 h - x_2) - p(x_1, x_2)|du_1 du_2 \tag{3.32}$$

$$+ \int_{\frac{x_2}{h}}^{1} \int_{\frac{x_1}{h}}^{1} K(u_1)K(u_2)|p(u_1 h - x_1, u_2 h - x_2) - p(x_1, x_2)|du_1 du_2 \tag{3.33}$$

$$\le 80Lh^2. \tag{3.34}$$

Therefore, we have $T_\mathcal{C} \le ch^6$.

Combining the analysis of $T_\mathcal{B}, T_\mathcal{C}$, and $T_\mathcal{I}$, we show that the mirror image kernel density estimator is free of boundary bias. Thus the desired result of Lemma 3.1 is proved. $\qquad\square$

### 3.1.3 Analyzing the Bias of the Entropy Estimator

**Lemma 3.2.** *Under Assumptions 2.1, 2.2, and 2.3, there exists a universal constant $C^*$ that does not depend on the true density $p$, such that*

$$\sup_{p \in \Sigma_\kappa(2, L)} \left| \mathbb{E}H\left(\widehat{p}_h\right) - H(p) \right| \le \frac{C^*}{\sqrt{n}}. \tag{3.35}$$

*Proof.* Recalling that $g(u) = u \log u$, by Taylor's theorem we have

$$g\left(\widehat{p}_h(x)\right) - g\left(p(x)\right) = \left(\log(p(x)) + 1\right) \cdot \left[\widehat{p}_h(x) - p(x)\right] + \frac{1}{2\xi(x)} \cdot \left[\widehat{p}_h(x) - p(x)\right]^2, \quad (3.36)$$

where $\xi(x)$ lies in between $\widehat{p}_h(x)$ and $p(x)$. It is obvious that $\kappa_1 \leq \xi(x) \leq \kappa_2$.

Let $\kappa$ be as defined in the statement of the theorem. Using Fubini's theorem, Hölder's inequality and the fact that the Lebesgue measure of $\mathcal{X}$ is 1, we have

$$\left|\mathbb{E}H\left(\widehat{p}_h\right) - H(p)\right| \tag{3.37}$$

$$= \left|\mathbb{E}\int_{\mathcal{X}} \left[g\left(\widehat{p}_h(x)\right) - g\left(p(x)\right)\right]dx\right| \tag{3.38}$$

$$= \left|\int_{\mathcal{X}} \mathbb{E}\left[g\left(\widehat{p}_h(x)\right) - g\left(p(x)\right)\right]dx\right| \tag{3.39}$$

$$\leq \left|\int_{\mathcal{X}} \left(\log(p(x)) + 1\right) \cdot \mathbb{E}\left[\widehat{p}_h(x) - p(x)\right]dx\right| + \left|\int_{\mathcal{X}} \frac{1}{2\xi(x)} \cdot \mathbb{E}\left[\widehat{p}_h(x) - p(x)\right]^2 dx\right|$$

$$\leq \kappa\sqrt{\int_{\mathcal{X}} \left[\mathbb{E}\widehat{p}_h(x) - p(x)\right]^2 dx} + \frac{1}{2\kappa_1} \cdot \int_{\mathcal{X}} \mathbb{E}\left[\widehat{p}_h(x) - p(x)\right]^2 dx \tag{3.40}$$

$$\leq \kappa\sqrt{\int_{\mathcal{X}} \left[\mathbb{E}\widetilde{p}_h(x) - p(x)\right]^2 dx} + \frac{1}{2\kappa_1} \cdot \int_{\mathcal{X}} \mathbb{E}\left[\widetilde{p}_h(x) - p(x)\right]^2 dx. \tag{3.41}$$

$$\leq c_1 h^2 + c_2 h^4 + \frac{c_3}{nh^2}. \tag{3.42}$$

The last inequality follows from standard results of kernel density estimation and Lemma 3.1, where $c_1, c_2, c_3$ are three constants. We get the desired result by setting $h \asymp n^{-1/4}$. $\qquad\square$

## 3.2 Analyzing the Variance Term

**Lemma 3.3.** *Under Assumptions 2.1, 2.2, and 2.3, we have,*

$$\sup_{p \in \Sigma_\kappa(2,L)} \mathbb{P}\left(\left|H\left(\widehat{p}_h\right) - \mathbb{E}H\left(\widehat{p}_h\right)\right| > \epsilon\right) \leq 2\exp\left(-\frac{n\epsilon^2}{32\kappa^2}\right). \tag{3.43}$$

*Proof.* Let $\widehat{p}'_h(x)$ be the kernel density estimator defined as in (2.6) but with the $j^{th}$ data point $x^j$ replaced by an arbitrary value $(x^j)'$. Since $g'(u) = \log u + 1$, by Assumption 2.1, we have $\max\left[\left|g'\left(\widehat{p}_h(x)\right)\right|, \left|g'\left(\widehat{p}'_h(x)\right)\right|\right] \leq \kappa$.

For notational simplicity, we write the product kernel as $K_2 = K \cdot K$. Using the mean-value theorem and the fact that $T_{\kappa_1, \kappa_2}(\cdot)$ is a contraction, we have

$$\sup_{x^1,\ldots,x^n,(x^j)'} \left|H\left(\widehat{p}_h\right) - H\left(\widehat{p}'_h\right)\right| \tag{3.44}$$

$$= \sup_{x^1,\ldots,x^n,(x^j)'} \left|\int_{\mathcal{X}} \left[g\left(\widehat{p}_h(x)\right) - g\left(\widehat{p}'_h(x)\right)\right]dx\right| \tag{3.45}$$

$$\leq \kappa \sup_{x^1,\ldots,x^n,(x^j)'} \int \left|\widehat{p}_h(x) - \widehat{p}'_h(x)\right| dx \tag{3.46}$$

$$= \kappa \sup_{x^1,\ldots,x^n,(x^j)'} \int_{\mathcal{X}} \left|T_{\kappa_1,\kappa_2}\left[\widetilde{p}_h(x)\right] - T_{\kappa_1,\kappa_2}\left[\widetilde{p}'_h(x)\right]\right| dx \tag{3.47}$$

$$\leq 4\kappa \sup_{x^1,\ldots,x^n,(x^j)'} \int_{\mathcal{X}} \left|\frac{1}{nh^2}K_2\left(\frac{x^j - x}{h}\right) - \frac{1}{nh^2}K_2\left(\frac{(x^j)' - x}{h}\right)\right| dx \tag{3.48}$$

$$\leq 8\kappa \sup_y \int_{\mathcal{X}} \frac{1}{nh^2}K_2\left(\frac{y - x}{h}\right) dx \tag{3.49}$$

$$\leq \frac{8\kappa}{n} \int K_2(u)du = \frac{8\kappa}{n}. \tag{3.50}$$

Therefore, using McDiarmaid's inequality [16], we get the desired inequality (3.43). The uniformity result holds since the constant does not depend on the true density $p$. $\qquad\square$

# 4 Application to Forest Density estimation

We apply the concentration inequality (2.10) to analyze an algorithm for learning high dimensional forest graph models [15]. In a forest density estimation problem, we observe $n$ data points $x^1, \ldots, x^n \in \mathbb{R}^d$ from a $d$-dimensional random vector $X$. We have two learning tasks: (i) we want to estimate an acyclic undirected graph $F = (V, E)$, where $V$ is the vertex set containing all the random variables and $E$ is the edge set such that an edge $(j, k) \in E$ if and only if the corresponding random variables $X_j$ and $X_k$ are conditionally independent given the other variables $X_{\setminus\{j,k\}}$; (ii) once we have an estimated graph $\widehat{F}$, we want to estimate the density function $p(x)$.

Using the negative log-likelihood loss, Liu et al. [15] show that the graph estimation problem can be recast as the problem of finding the maximum weight spanning forest for a weighted graph, where the weight of the edge connecting nodes $j$ and $k$ is $I(X_j; X_k)$, the mutual information between these two variables. Empirically, we replace $I(X_j; X_k)$ by its estimate $\widehat{I}(X_j; X_k)$ from (2.9). The forest graph can be obtained by the Chow-Liu algorithm [3, 13], which is an iterative algorithm. At each iteration the algorithm adds an edge connecting that pair of variables with maximum mutual information among all pairs not yet visited by the algorithm, if doing so does not form a cycle. When stopped early, after $s < d - 1$ edges have been added, it yields the best $s$-edge weighted forest. Once a forest graph $\widehat{F} = (V, \widehat{E})$ is estimated, we propose to estimate the forest density as

$$\widehat{p}_{\widehat{F}}(x) = \prod_{(j,k)\in\widehat{E}} \frac{\widetilde{p}_{h_2}(x_j, x_k)}{\widetilde{p}_{h_2}(x_j)\widetilde{p}_{h_2}(x_k)} \cdot \prod_{u\in\widehat{U}} \widetilde{p}_{h_2}(x_u) \cdot \prod_{\ell\in V\setminus\widehat{U}} \widetilde{p}_{h_1}(x_\ell), \tag{4.1}$$

where $\widehat{U}$ is the set of isolated vertices in the estimated forest $\widehat{F}$. Our estimator is different from the estimator proposed by [15]—once the graph $\widehat{F}$ is given, we treat the isolated variables differently than the connected variables. As will be shown in Theorem 4.2, such a choice leads to minimax optimal forest density estimation, while the obtained rate from [15] is suboptimal.

Let $\mathcal{F}_d^s$ denote the set of forest graphs with $d$ nodes and no more than $s$ edges. Let $D(\cdot\|\cdot)$ be the Kullback-Leibler divergence. We define the $s$-oracle forest $F_s^* := (V, E^*)$ and its corresponding oracle density estimator $p_{F^*}$ to be

$$F_s^* = \arg\min_{F\in\mathcal{F}_d^s} D(p\|p_F) \quad \text{and} \quad p_{F^*} := \prod_{(j,k)\in E^*} \frac{p(x_j, x_k)}{p(x_j, x_k)} \prod_{\ell\in V} p(x_\ell). \tag{4.2}$$

Let $\Sigma_\kappa(2, L)$ be defined as in Assumption (2.1). We define a density class $\mathcal{P}_\kappa$ as

$$\mathcal{P}_\kappa := \big\{ p : p \text{ is a } d\text{-dimensional density with } p(x_j, x_k) \in \Sigma_\kappa(2, L) \text{ for any } j \neq k \big\}. \tag{4.3}$$

The next two theorems show that the above forest density estimation procedure is minimax optimal for both graph recovery and density estimation. Their proofs are provided in a technical report [14].

**Theorem 4.1** (**Graph Recovery**). *Let $\widehat{F}$ be the estimated $s$-edge forest graph using the Chow-Liu algorithm. Under the same condition as Theorem 12 in [15], If we choose $h \asymp n^{-1/4}$ for the mutual information estimator in* (2.9)*, then*

$$\sup_{p\in\mathcal{P}_\kappa} \mathbb{P}\left(\widehat{F} \neq F_s^*\right) = O\left(\sqrt{\frac{s}{n}}\right) \quad \text{whenever} \quad \frac{\log d}{n} \to 0. \tag{4.4}$$

**Theorem 4.2** (**Density Estimation**). *Once the $s$-edge forest graph $\widehat{F}$ as in Theorem 4.1 has been obtained, we calculate the density estimator* (B.1) *by choosing $h_1 \asymp n^{-1/5}$ and $h_2 \asymp n^{-1/6}$. Then,*

$$\sup_{p\in\mathcal{P}_\kappa} \mathbb{E} \int_{\mathcal{X}} \left|\widehat{p}_{\widehat{F}}(x) - p_{F^*}(x)\right| dx \leq C \cdot \sqrt{\frac{s}{n^{2/3}} + \frac{d-s}{n^{4/5}}}. \tag{4.5}$$

# 5 Discussions and Conclusions

Theorem 4.1 allows $d$ to increase exponentially fast as $n$ increases and still guarantees graph recovery consistency. Theorem 4.2 provides the rate of convergence for the $L_1$-risk. The obtained rate is minimax optimal over the class $\mathcal{P}_\kappa$. The term $sn^{-2/3}$ corresponds to the price paid to estimate bivariate densities; while the term $(d-s)n^{-4/5}$ corresponds to the price paid to estimate univariate densities. In this way, we see that the exponential concentration inequality for Shannon mutual information leads to significantly improved theoretical analysis of the forest density estimation, in terms of both graph estimation and density estimation. This research was supported by NSF grant IIS-1116730 and AFOSR contract FA9550-09-1-0373.

# References

[1] Ibrahim A. Ahmad and Pi-Erh Lin. A nonparametric estimation of the entropy for absolutely continuous distributions (corresp.). *IEEE Transactions on Information Theory*, 22(3):372–375, 1976.

[2] J Beirlant, E J Dudewicz, L Györfi, and E C Van Der Meulen. Nonparametric entropy estimation: An overview. *International Journal of Mathematical and Statistical Sciences*, 6(1):17–39, 1997.

[3] C. Chow and C. Liu. Approximating discrete probability distributions with dependence trees. *Information Theory, IEEE Transactions on*, 14(3):462–467, 1968.

[4] Thomas M. Cover and Joy A. Thomas. *Elements of Information Theory*. Wiley, 1991.

[5] Paul P. B. Eggermont and Vincent N. LaRiccia. Best asymptotic normality of the kernel density entropy estimator for smooth densities. *IEEE Transactions on Information Theory*, 45(4):1321–1326, 1999.

[6] A. Gretton, R. Herbrich, and A. J. Smola. The kernel mutual information. In *Acoustics, Speech, and Signal Processing, 2003. Proceedings.(ICASSP'03). 2003 IEEE International Conference on*, volume 4, pages IV–880. IEEE, 2003.

[7] Peter Hall and Sally Morton. On the estimation of entropy. *Annals of the Institute of Statistical Mathematics*, 45(1):69–88, 1993.

[8] A. O. Hero III, B. Ma, O. J. J. Michel, and J. Gorman. Applications of entropic spanning graphs. *Signal Processing Magazine, IEEE*, 19(5):85–95, 2002.

[9] Harry Joe. Estimation of entropy and other functionals of a multivariate density. *Annals of the Institute of Statistical Mathematics*, 41(4):683–697, December 1989.

[10] M. C. Jones, M. C. Linton, and J. P. Nielsen. A simple bias reduction method for density estimation. *Biometrika*, 82(2):327–338, 1995.

[11] Shiraj Khan, Sharba Bandyopadhyay, Auroop R. Ganguly, Sunil Saigal, David J. Erickson, Vladimir Protopopescu, and George Ostrouchov. Relative performance of mutual information estimation methods for quantifying the dependence among short and noisy data. *Phys. Rev. E*, 76:026209, Aug 2007.

[12] Alexander Kraskov, Harald Stögbauer, and Peter Grassberger. Estimating mutual information. *Physical review. E, Statistical, nonlinear, and soft matter physics*, 69(6 Pt 2), June 2004.

[13] Joseph B. Kruskal. On the shortest spanning subtree of a graph and the traveling salesman problem. *Proceedings of the American Mathematical Society*, 7(1):48–50, 1956.

[14] Han Liu, John Lafferty, and Larry Wasserman. Optimal forest density estimation. *Technical Report*, 2012.

[15] Han Liu, Min Xu, Haijie Gu, Anupam Gupta, John D. Lafferty, and Larry A. Wasserman. Forest density estimation. *Journal of Machine Learning Research*, 12:907–951, 2011.

[16] C. McDiarmid. On the method of bounded differences. In *Surveys in Combinatorics*, number 141 in London Mathematical Society Lecture Note Series, pages 148–188. Cambridge University Press, August 1989.

[17] XuanLong Nguyen, Martin J. Wainwright, and Michael I. Jordan. Estimating divergence functionals and the likelihood ratio by convex risk minimization. *IEEE Transactions on Information Theory*, 56(11):5847–5861, 2010.

[18] D. Pál, B. Póczos, and C. Szepesvári. Estimation of Rényi entropy and mutual information based on generalized nearest-neighbor graphs. *Arxiv preprint arXiv:1003.1954*, 2010.

[19] L. Paninski. Estimation of entropy and mutual information. *Neural Computation*, 15(6):1191–1253, 2003.

[20] Liam Paninski and Masanao Yajima. Undersmoothed kernel entropy estimators. *IEEE Transactions on Information Theory*, 54(9):4384–4388, 2008.

[21] Barnabás Póczos and Jeff G. Schneider. Nonparametric estimation of conditional information and divergences. *Journal of Machine Learning Research - Proceedings Track*, 22:914–923, 2012.

[22] B. W. Silverman. *Density Estimation for Statistics and Data Analysis*. Chapman and Hall. New York, NY, 1986.

[23] A.B. Tsybakov and van den Meulen. *Root-n Consistent Estimators of Entropy for Densities with Unbounded Support*, volume 23. Universite catholique de Louvain,Institut de statistique, 1994.

[24] Marc M. Van Hulle. Edgeworth approximation of multivariate differential entropy. *Neural Comput.*, 17(9):1903–1910, September 2005.

[25] O Vasicek. A test for normality based on sample entropy. *Journal of the Royal Statistical Society Series B*, 38(1):54–59, 1976.

[26] Ven Es Bert. Estimating functionals related to a density by a class of statistics based on spacings. *Scandinavian Journal of Statistics*, 19(1):61–72, 1992.

